# Mapping a manifold of perceptual observations

**Joshua B. Tenenbaum**
Department of Brain and Cognitive Sciences
Massachusetts Institute of Technology, Cambridge, MA 02139
jbt@psyche.mit.edu

## Abstract

Nonlinear dimensionality reduction is formulated here as the problem of trying to find a Euclidean feature-space embedding of a set of observations that preserves as closely as possible their intrinsic metric structure – the distances between points on the observation manifold as measured along geodesic paths. Our *isometric feature mapping* procedure, or isomap, is able to reliably recover low-dimensional nonlinear structure in realistic perceptual data sets, such as a manifold of face images, where conventional global mapping methods find only local minima. The recovered map provides a canonical set of globally meaningful features, which allows perceptual transformations such as interpolation, extrapolation, and analogy – highly nonlinear transformations in the original observation space – to be computed with simple linear operations in feature space.

## 1 Introduction

In psychological or computational research on perceptual categorization, it is generally taken for granted that the perceiver has *a priori* access to a representation of stimuli in terms of some perceptually meaningful features that can support the relevant classification. However, these features will be related to the raw sensory input (e.g. values of retinal activity or image pixels) only through a very complex transformation, which must somehow be acquired through a combination of evolution, development, and learning. Fig. 1 illustrates the feature-discovery problem with an example from visual perception. The set of views of a face from all possible viewpoints is an extremely high-dimensional data set when represented as image arrays in a computer or on a retina; for example, 32 x 32 pixel grey-scale images can be thought of as points in a 1,024-dimensional observation space. The perceptually meaningful structure of these images, however, is of much lower dimensionality; all of the images in Fig. 1 lie on a two-dimensional manifold parameterized by viewing angle. A perceptual system that discovers this manifold structure has learned a model of the appearance of this face that will support a wide range of recognition, classification, and imagery tasks (some demonstrated in Fig. 1), despite the absence of any prior physical knowledge about three-dimensional object geometry, surface texture, or illumination conditions.

Learning a manifold of perceptual observations is difficult because these observations

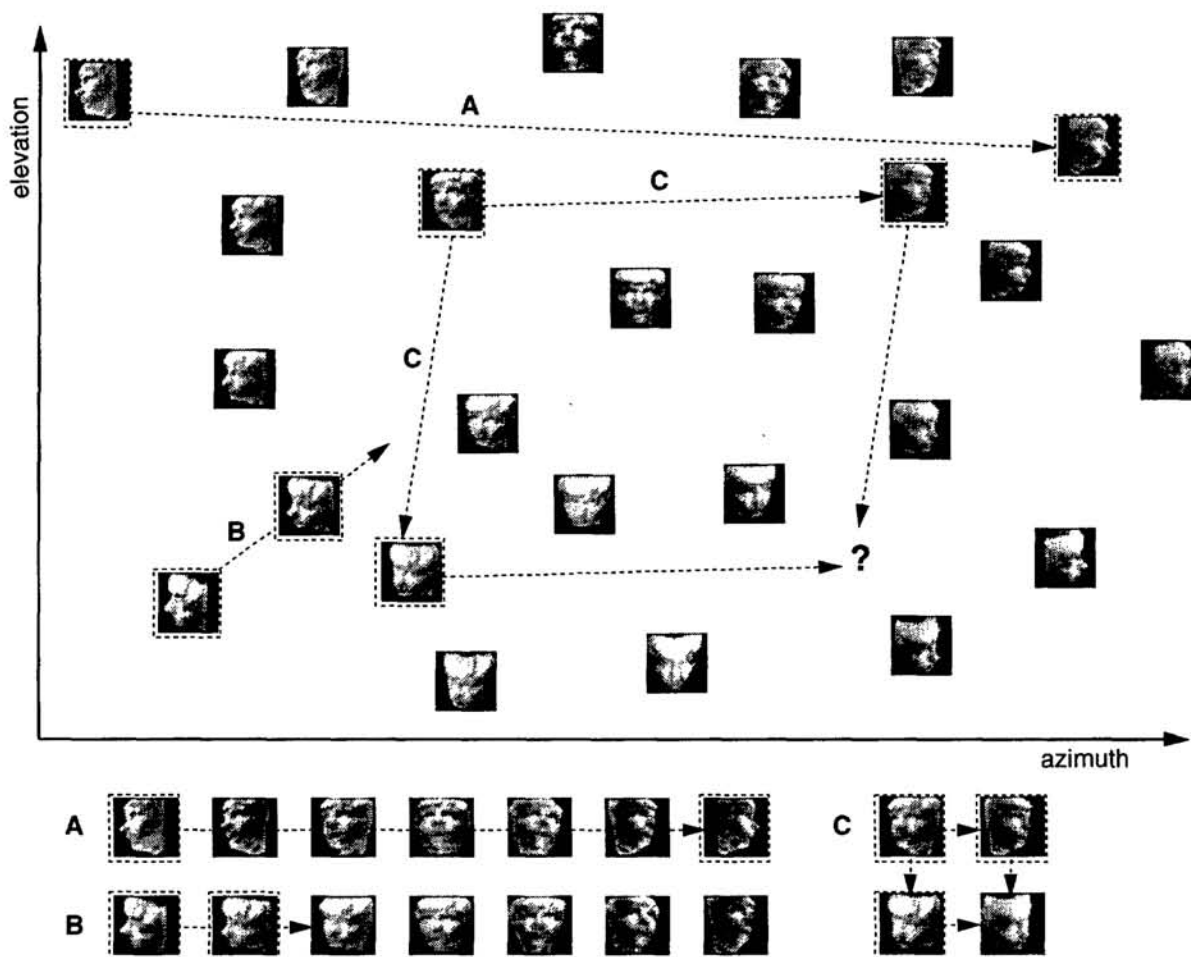

Figure 1: Isomap recovers a global topographic map of face images varying in two viewing angle parameters, azimuth and elevation. Image interpolation (A), extrapolation (B), and analogy (C) can then be carried out by linear operations in this feature space.

usually exhibit significant nonlinear structure. Fig. 2A provides a simplified version of this problem. A flat two-dimensional manifold has been nonlinearly embedded in a three-dimensional observation space, [1] and must be "unfolded" by the learner. For linearly embedded manifolds, principal component analysis (PCA) is guaranteed to discover the dimensionality of the manifold and produce a compact representation in the form of an orthonormal basis. However, PCA is completely insensitive to the higher-order, nonlinear structure that characterizes the points in Fig. 2A or the images in Fig. 1.

Nonlinear dimensionality reduction – the search for intrinsically low-dimensional structures embedded nonlinearly in high-dimensional observations – has long been a goal of computational learning research. The most familiar nonlinear techniques, such as the self-organizing map (SOM; Kohonen, 1988), the generative topographic mapping (GTM; Bishon, Svensen, & Williams, 1998), or autoencoder neural networks (DeMers & Cottrell, 1993), try to generalize PCA by discovering a single *global* low-dimensional nonlinear model of the observations. In contrast, *local* methods (Bregler & Omohundro, 1995; Hinton, Revow, & Dayan, 1995) seek a set of low-dimensional models, usually linear and hence valid only for a limited range of data. When appropriate, a single global model is

[1]Given by $x_1 = z_1 \cos(z_1)$, $x_2 = z_1 \sin(z_1)$, $x_3 = z_2$, for $z_1 \in [3\pi/2, 9\pi/2], z_2 \in [0, 15]$.

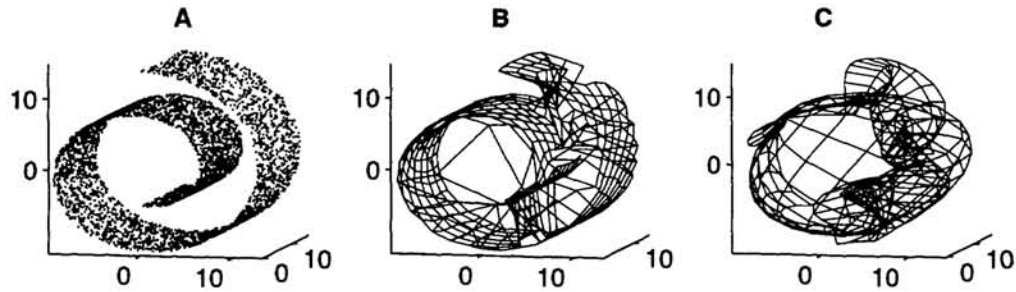

Figure 2: A nonlinearly embedded manifold may create severe local minima for "top-down" mapping algorithms. (A) Raw data. (B) Best SOM fit. (C) Best GTM fit.

more revealing and useful than a set of local models. However, local linear methods are in general far more computationally efficient and reliable than global methods.

For example, despite the visually obvious structure in Fig. 2A, this manifold was not successfuly modeled by either of two popular global mapping algorithms, SOM (Fig. 2B) and GTM (Fig. 2C), under a wide range of parameter settings. Both of these algorithms try to fit a grid of predefined (usually two-dimensional) topology to the data, using greedy optimization techniques that first fit the large-scale (linear) structure of the data, before making small-scale (nonlinear) refinements. The coarse structure of such "folded" data sets as Fig. 2A hides their nonlinear structure from greedy optimizers, virtually ensuring that top-down mapping algorithms will become trapped in highly suboptimal solutions.

Rather than trying to force a predefined map onto the data manifold, this paper shows how a perceptual system may map a set of observations in a "bottom-up" fashion, by first learning the topological structure of the manifold (as in Fig. 3A) and only then learning a metric map of the data (as in Fig. 3C) that respects this topology. The next section describes the goals and steps of the mapping procedure, and subsequent sections demonstrate applications to two challenging learning tasks: recovering a five-dimensional manifold embedded nonlinearly in 50 dimensions, and recovering the manifold of face images depicted in Fig. 1.

## 2   Isometric feature mapping

We assume our data lie on an unknown manifold $M$ embedded in a high-dimensional observation space $X$. Let $x^{(i)}$ denote the coordinates of the $i$th observation. We seek a mapping $f : X \to Y$ from the observation space $X$ to a low-dimensional Euclidean feature space $Y$ that preserves as well as possible the intrinsic metric structure of the observations, i.e. the distances between observations as measured along geodesic (locally shortest) paths of $M$. The *isometric feature mapping*, or isomap, procedure presented below generates an implicit description of the mapping $f$, in terms of the corresponding feature points $y^{(i)} = f(x^{(i)})$ for sufficiently many observations $x^{(i)}$. Explicit parametric descriptions of $f$ or $f^{-1}$ can be found with standard techniques of function approximation (Poggio & Girosi, 1990) that interpolate smoothly between the known corresponding pairs $\{x^{(i)}, y^{(i)}\}$.

A Euclidean map of the data's intrinsic geometry has several important properties. First, intrinsically similar observations should map to nearby points in feature space, supporting efficient similarity-based classification and informative visualization. Moreover, the geodesic paths of the manifold, which are highly nonlinear in the original observation space, should map onto straight lines in feature space. Then perceptually natural transformations along these paths, such as the interpolation, extrapolation and analogy demonstrated in Figs. 1A-C, may be computed by trivial linear operations in feature space.

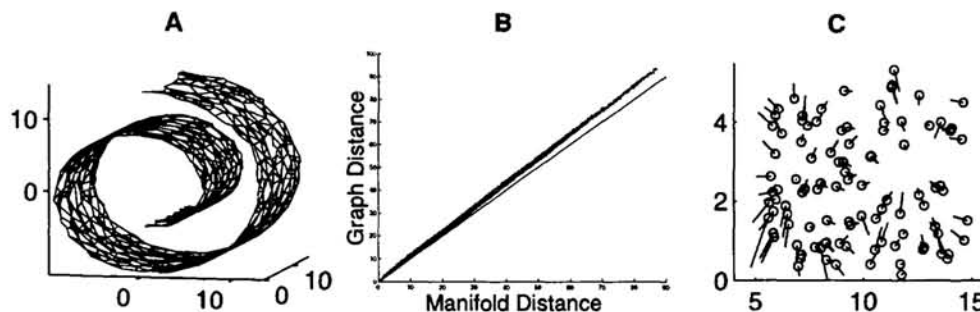

Figure 3: The results of the three-step isomap procedure. (A) Discrete representation of manifold in Fig. 2A. (B) Correlation between measured graph distances and true manifold distances. (C) Correspondence of recovered two-dimensional feature points $\{y_1, y_2\}$ (circles) with original generating vectors $\{z_1, z_2\}$ (line ends).

The isomap procedure consists of three main steps, each of which might be carried out by more or less sophisticated techniques. The crux of isomap is finding an efficient way to compute the true geodesic distance between observations, given only their Euclidean distances in the high-dimensional observation space. Isomap assumes that distance between points in observation space is an accurate measure of manifold distance only locally and must be integrated over paths on the manifold to obtain global distances. As preparation for computing manifold distances, we first construct a discrete representation of the manifold in the form of a topology-preserving network (Fig. 3A). Given this network representation, we then compute the shortest-path distance between any two points in the network using dynamic programming. This polynomial-time computation provides a good approximation to the actual manifold distances (Fig. 3B) without having to search over all possible paths in the network (let alone the infinitely many paths on the unknown manifold!). Finally, from these manifold distances, we construct a global geometry-preserving map of the observations in a low-dimensional Euclidean space, using multidimensional scaling (Fig. 3C). The implementation of this procedure is detailed below.

**Step 1: Discrete representation of manifold (Fig. 3A).** From the input data of $n$ observations $\{x^{(1)}, \ldots, x^{(n)}\}$, we randomly select a subset of $r$ points to serve as the nodes $\{g^{(1)}, \ldots, g^{(r)}\}$ of the topology-preserving network. We then construct a graph $G$ over these nodes by connecting $g^{(i)}$ and $g^{(j)}$ if and only if there exists at least one $\mathbf{x}^{(k)}$ whose two closest nodes (in observation space) are $g^{(i)}$ and $g^{(j)}$ (Martinetz & Schulten, 1994). The resulting graph for the data in Fig. 2A is shown in Fig. 3A (with $n = 10^4, r = 10^3$). This graph clearly respects the topology of the manifold far better than the best fits with SOM (Fig. 2B) or GTM (Fig. 2C). In the limit of infinite data, the graph thus produced converges to the Delaunay triangulation of the nodes, restricted to the data manifold (Martinetz & Schulten, 1994). In practice, $n = 10^4$ data points have proven sufficient for all examples we have tried. This number may be reduced significantly if we know the dimensionality $d$ of the manifold, but here we assume no *a priori* information about dimensionality. The choice of $r$, the number of nodes in $G$, is the only free parameter in isomap. If $r$ is too small, the shortest-path distances between nodes in $G$ will give a poor approximation to their true manifold distance. If $r$ is too big (relative to $n$), $G$ will be missing many appropriate links (because each data point $\mathbf{x}^{(i)}$ contributes at most one link). In practice, choosing a satisfactory $r$ is not difficult – all three examples presented in this paper use $r = n/10$, the first value tried. I am currently exploring criteria for selecting the optimal value $r$ based on statistical arguments and dimensionality considerations.

**Step 2: Manifold distance measure (Fig. 3B).** We first assign a weight to each link $w_{ij}$ in the graph $G$, equal to $d_X^{ij} = \|x^{(i)} - x^{(j)}\|$, the Euclidean distance between nodes $i$ and $j$ in the observation space $X$. The length of a path in $G$ is defined to be the sum of link weights along that path. We then compute the geodesic distance $d_G^{ij}$ (i.e. shortest path length) between all pairs of nodes $i$ and $j$ in $G$, using Floyd's $O(r^3)$ algorithm (Foster, 1995). Initialize $d_G^{ij} = d_X^{ij}$ if nodes $i$ and $j$ are connected

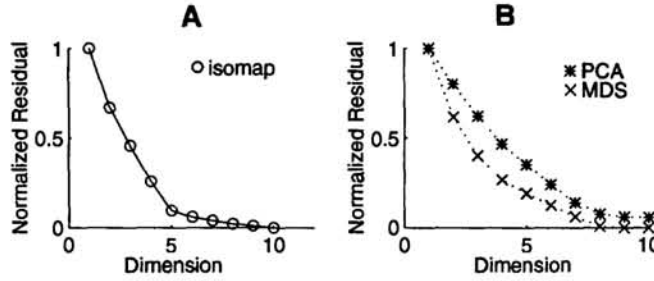

Figure 4: Given a 5–dimensional manifold embedded nonlinearly in a 50–dimensional space, isomap identifies the intrinsic dimensionality (A), while PCA and MDS alone do not (B).

and $\infty$ otherwise. Then for each node $k$, set each $d_G^{ij} = \min(d_G^{ij}, d_G^{ik} + d_G^{kj})$. Fig. 3B plots the distances $d_G^{ij}$ computed between nodes $i$ and $j$ in the graph of Fig. 3A versus their actual manifold distances $d_M^{ij}$. Note that the correlation is almost perfect ($R > .99$), but $d_G^{ij}$ tends to overestimate $d_M^{ij}$ by a constant factor due to the discretization introduced by the graph. As the density of observations increases, so does the possible graph resolution. Thus, in the limit of infinite data, the graph-based approximation to manifold distance may be made arbitrarily accurate.

**Step 3: Isometric Euclidean embedding (Fig. 3C).** We use ordinal multidimensional scaling (MDS; Cox & Cox, 1994; code provided by Brian Ripley), also called "nonmetric" MDS, to find a $k$-dimensional Euclidean embedding that preserves as closely as possible the graph distances $d_G^{ij}$. In contrast to classical "metric" MDS, which explicitly tries to preserve distances, ordinal MDS tries to preserve only the rank ordering of distances. MDS finds a configuration of $k$-dimensional feature vectors $\{y^{(1)}, \ldots, y^{(r)}\}$, corresponding to the high-dimensional observations $\{x^{(1)}, \ldots, x^{(r)}\}$, that minimizes the stress function,

$$S = \min_{\hat{d}_G^{ij}} \sqrt{\frac{\sum_{i<j}(d_Y^{ij} - \hat{d}_G^{ij})^2}{\sum_{i<j}(d_Y^{ij})^2}}. \tag{1}$$

Here $d_Y^{ij} = \|y^{(i)} - y^{(j)}\|$, the Euclidean distance between feature vectors $i$ and $j$, and the $\hat{d}_G^{ij}$ are some monotonic transformation of the graph distances $d_G^{ij}$. We use ordinal MDS because it is less sensitive to noisy estimates of manifold distance. Moreover, when the number of points scaled is large enough (as it is in all our examples), ordinal constraints alone are sufficient to reconstruct a precise metric map. Fig. 3C shows the projections of 100 random points on the manifold in Fig. 2A onto a two-dimensional feature space computed by MDS from the graph distances output by step 2 above. These points are in close correspondence (after rescaling) with the original two-dimensional vectors used to generate the manifold (see note 1), indicating that isomap has successfully unfolded the manifold onto a 2-dimensional Euclidean plane.

## 3   Example 1: Five-dimensional manifold

This section demonstrates isomap's ability to discover and model a noisy five-dimensional manifold embedded within a 50-dimensional space. As the dimension of the manifold increases beyond two, SOM, GTM, and other constrained clustering approaches become impractical due to the exponential proliferation of cluster centers. Isomap, however, is quite practical for manifolds of moderate dimensionality, because the estimates of manifold distance for a fixed graph size degrade gracefully as dimensionality increases. Moreover, isomap is able to automatically discover the intrinsic dimensionality of the data, while conventional methods must be initialized with a fixed dimensionality.

We consider a 5-dimensional manifold parameterized by $\{z_1, \ldots, z_5\} \in [0, 4]^5$. The first 10 of 50 observation dimensions were determined by nonlinear functions of these parameters. [2]

Low-amplitude gaussian noise (4-5% of variance) was added to each of these dimensions, and the remaining 40 dimensions were set to pure noise of similar variance. The isomap procedure applied to this data ($n = 10^4, r = 10^3$) correctly recognized its intrinsic five-dimensionality, as indicated by the sharp decrease of stress (see Eq. 1) for embedding dimensions up to 5 and only gradual decrease thereafter (Fig. 4A). In contrast, both PCA and raw MDS (using distances in observation space rather than manifold distances) identify the 10-dimensional linear subspace containing the data, but show no sensitivity to the underlying five-dimensional manifold (Fig. 4B).

## 4   Example 2: Two-dimensional manifold of face images

This section illustrates the performance of isomap on the two-dimensional manifold of face images shown in Fig. 1. To generate this map, 32 x 32-pixel images of a face were first rendered in MATLAB in many different poses (azimuth $\in [-90°, 90°]$, elevation $\in [-10°, 10°]$), using a 3-D range image of an actual head and a combination of lambertian and specular reflectance models. To save computation, the data ($n = 10^4$ images) were first reduced to 60 principal components and then submitted to isomap ($r = 10^3$). The plot of stress $S$ vs. dimension indicated a dimensionality of two (even more clearly than Fig. 4A). Fig. 1 shows the two-dimensional feature space that results from applying MDS to the computed graph distances, with 25 face images placed at their corresponding points in feature space. Note the clear topographic representation of similar views at nearby feature points. The principal axes of the feature space can be identified as the underlying viewing angle parameters used to generate the data. The correlations of the two isomap dimensions with the two pose angles are $R = .99$ and $R = .95$ respectively. No other global mapping procedure tried (PCA, MDS, SOM, GTM) produced interpretable results for these data.

The human visual system's implicit knowledge of an object's appearance is not limited to a representation of view similarity, and neither is isomap's. As mentioned in Section 2, an isometric feature map also supports analysis and manipulation of data, as a consequence of mapping geodesics of the observation manifold to straight lines in feature space. Having found a number of corresponding pairs $\{x^{(i)}, y^{(i)}\}$ of images $x^{(i)}$ and feature vectors $y^{(i)}$, it is easy to learn an explicit inverse mapping $f^{-1} : Y \to X$ from low-dimensional feature space to high-dimensional observation space, using generic smooth interpolation techniques such as generalized radial basis function (GRBF) networks (Poggio & Girosi, 1990). All images in Fig. 1 have been synthesized from such a mapping. [3]

Figs. 1A-C show how learning this inverse mapping allows interpolation, extrapolation, and analogy to be carried out using only linear operations. We can interpolate between two images $x^{(1)}$ and $x^{(2)}$ by synthesizing a sequence of images along their connecting line $(y^{(2)} - y^{(1)})$ in feature space (Fig. 1A). We can extrapolate the transformation from one image to another and far beyond, by following the line to the edge of the manifold (Fig. 1B). We can map the transformation between two images $x^{(1)}$ and $x^{(2)}$ onto an analogous transformation of another image $x^{(3)}$, by adding the transformation vector $(y^{(2)} - y^{(1)})$ to $y^{(3)}$ and synthesizing a new image at the resulting feature coordinates (Fig. 1C).

A number of authors (Bregler & Omohundro, 1995; Saul & Jordan, 1997; Beymer & Poggio, 1995) have previously shown how learning from examples allows sophisticated

image manipulations to be carried out efficiently. However, these approaches do not support as broad a range of transformations as isomap does, because of their use of only locally valid models and/or the need to compute special-purpose image features such as optical flow. See Tenenbaum (1997) for further discussion, as well as examples of isomap applied to more complex manifolds of visual observations.

## 5   Conclusions

The essence of the isomap approach to nonlinear dimensionality reduction lies in the novel problem formulation: to seek a low-dimensional Euclidean embedding of a set of observations that captures their intrinsic similarities, as measured along geodesic paths of the observation manifold. Here I have presented an efficient algorithm for solving this problem and shown that it can discover meaningful feature-space models of manifolds for which conventional "top-down" approaches fail. As a direct consequence of mapping geodesics to straight lines in feature space, isomap learns a representation of perceptual observations in which it is easy to perform interpolation and other complex transformations. A negative consequence of this strong problem formulation is that isomap will not be applicable to every data manifold. However, as with the classic technique of PCA, we can state clearly the general class of data for which isomap is appropriate – manifolds with no "holes" and no intrinsic curvature – with a guarantee that isomap will succeed on data sets from this class, given enough samples from the manifold. Future work will focus on generalizing this domain of applicability to allow for manifolds with more complex topologies and significant curvature, as would be necessary to model certain perceptual manifolds such as the complete view space of an object.

## Acknowledgements

Thanks to M. Bernstein, W. Freeman, S. Gilbert, W. Richards, and Y. Weiss for helpful discussions. The author is a Howard Hughes Medical Institute Predoctoral Fellow.

## Footnotes

[2] $x_1 = \cos(\pi z_1)$, $x_2 = \sin(\pi z_1)$, $x_3 = \cos(\frac{2\pi}{3}z_1)$, $x_4 = \sin(\frac{2\pi}{3}z_1)$, $x_5 = \cos(\frac{\pi}{3}z_1)$, $x_6 = \sin(\frac{\pi}{3}z_1)$, $x_7 = z_2\cos^2(\frac{\pi}{32}z_1) + z_3\sin^2(\frac{\pi}{32}z_1)$, $x_8 = z_2\sin^2(\frac{\pi}{32}z_1) + z_3\cos^2(\frac{\pi}{32}z_1)$, $x_9 = z_4\cos^2(\frac{\pi}{32}z_1) + z_5\sin^2(\frac{\pi}{32}z_1)$, $x_{10} = z_4\sin^2(\frac{\pi}{32}z_1) + z_5\cos^2(\frac{\pi}{32}z_1)$.

[3]The map from feature vectors to images was learned by fitting a GRBF net to 1000 corresponding points in both spaces. Each point corresponds to a node in the graph G used to measure manifold distance, so the feature-space distances required to fit the GRBF net are given (approximately) by the graph distances $d_G^{ij}$ computed in step 2 of isomap. A subset $C$ of $m = 300$ points were randomly chosen as RBF centers, and the standard deviation of the RBFs was set equal to $\max_{i,j \in C} d_G^{ij}/\sqrt{2m}$ (as prescribed by Haykin, 1994).

## References

Beymer, D. & Poggio, T. (1995). Representations for visual learning, *Science* **272**, 1905.

Bishop, C., Svensen, M., & Williams, C. (1998). GTM: The generative topographic mapping. *Neural Computation* **10(1)**.

Bregler, C. & Omohundro, S. (1995). Nonlinear image interpolation using manifold learning. *NIPS 7*. MIT Press.

Cox, T. & Cox, M. (1994). *Multidimensional scaling*. Chapman & Hall.

DeMers, D. & Cottrell, G. (1993). Nonlinear dimensionality reduction. *NIPS 5*. Morgan Kauffman.

Foster, I. (1995). *Designing and building parallel programs*. Addison-Wesley.

Haykin, S. (1994). *Neural Networks: A Comprehensive Foundation*. Macmillan.

Hinton, G., Revow, M., & Dayan, P. (1995). Recognizing handwritten digits using mixtures of linear models. *NIPS 7*. MIT Press.

Kohonen, T. (1988). *Self-Organization and Associative Memory*. Berlin: Springer.

Martinetz, T. & Schulten, K. (1994). Topology representing networks. *Neural Networks* **7**, 507.

Poggio, T. & Girosi, F. (1990). Networks for approximation and learning. *Proc. IEEE* **78**, 1481.

Saul, L. & Jordan, M. (1997). A variational principle for model-based morphing. *NIPS 9*. MIT Press.

Tenenbaum, J. (1997). Unsupervised learning of appearance manifolds. Manuscript submitted.